# Co-regularization Based Semi-supervised Domain Adaptation

**Hal Daumé III**
Department of Computer Science
University of Maryland CP, MD, USA
`hal@umiacs.umd.edu`

**Abhishek Kumar**
Department of Computer Science
University of Maryland CP, MD, USA
`abhishek@umiacs.umd.edu`

**Avishek Saha**
School Of Computing
University of Utah, UT, USA
`avishek@cs.utah.edu`

## Abstract

This paper presents a co-regularization based approach to semi-supervised domain adaptation. Our proposed approach (EA++) builds on the notion of augmented space (introduced in EASYADAPT (EA) [1]) and harnesses unlabeled data in target domain to further assist the transfer of information from *source* to *target*. This semi-supervised approach to domain adaptation is extremely simple to implement and can be applied as a pre-processing step to any supervised learner. Our theoretical analysis (in terms of Rademacher complexity) of EA and EA++ show that the hypothesis class of EA++ has lower complexity (compared to EA) and hence results in tighter generalization bounds. Experimental results on sentiment analysis tasks reinforce our theoretical findings and demonstrate the efficacy of the proposed method when compared to EA as well as few other representative baseline approaches.

## 1 Introduction

A domain adaptation approach for NLP tasks, termed EASYADAPT (EA), augments the *source domain* feature space using features from labeled data in *target domain* [1]. EA is simple, easy to extend and implement as a preprocessing step and most importantly is agnostic of the underlying classifier. However, EA requires labeled data in both source and target, and hence applies to fully supervised domain adaptation settings *only*. In this paper, [1] we propose a semi-supervised [2] approach to leverage unlabeled data for EASYADAPT (which we call EA++) and theoretically, as well as empirically, demonstrate its superior performance over EA.

There exists prior work on supervised domain adaptation (and multi-task learning) that can be related to EASYADAPT. An algorithm for multi-task learning using shared parameters was proposed for multi-task regularization [3] wherein each task parameter was represented as sum of a mean parameter (that stays same for all tasks) and its deviation from this mean. SVMs were used as the base classifiers and the algorithm was formulated in the standard SVM dual optimization setting. Subsequently, this framework was extended to online multi-domain setting in [4]. Prior work on semi-supervised approaches to domain adaptation also exists in literature. Extraction of specific features from the available dataset was proposed [5, 6] to facilitate the task of domain adaptation. Co-adaptation [7], a combination of co-training and domain adaptation, can also be considered as a semi-supervised approach to domain adaptation. A semi-supervised EM algorithm for domain adaptation was proposed in [8]. Similar to graph based semi-supervised approaches, a label propagation method was proposed [9] to facilitate domain adaptation. Domain Adaptation Machine (DAM) [10] is a semi-supervised extension of SVMs for domain adaptation and presents extensive empirical results. Nevertheless, in almost all of the above cases, the proposed methods either use specifics of the datasets or are customized for some particular base classifier and hence it is not clear how the proposed methods can be extended to other existing classifiers.

As mentioned earlier, EA is remarkably general in the sense that it can be used as a pre-processing step in conjunction with any base classifier. However, one of the prime limitations of EA is its incapability to leverage unlabeled data. Given its simplicity and generality, it would be interesting to extend EA to semi-supervised settings. In this paper, we propose EA++, a co-regularization based semi-supervised extension to EA. We also present Rademacher complexity based generalization bounds for EA and EA++. Our generalization bounds also apply to the approach proposed in [3] for domain adaptation setting, where we are only concerned with the error on target domain. The closest to our work is a recent paper [11] that theoretically analyzes EASYADAPT. Their paper investigates the necessity to combine *supervised* and *unsupervised* domain adaptation (which the authors refer to as *labeled* and *unlabeled* adaptation frameworks, respectively) and analyzes the combination using mistake bounds (which is limited to perceptron-based online scenarios). In addition, their work points out that EASYADAPT is limited to only supervised domain adaptation. On the contrary, our work extends EASYADAPT to semi-supervised settings and presents generalization bound based theoretical analysis which specifically demonstrate why EA++ is better than EA.

## 2 Background

In this section, we introduce notations and provide a brief overview of EASYADAPT [1].

### 2.1 Problem Setup and Notations

Let $\mathcal{X} \subset \mathbb{R}^d$ denote the instance space and $\mathcal{Y} = \{-1, +1\}$ denote the label space. Let $\mathcal{D}_s(x, y)$ be the source distribution and $\mathcal{D}_t(x, y)$ be the target distribution. We have a set of source labeled examples $L_s(\sim \mathcal{D}_s(x, y))$ and a set of target labeled examples $L_t(\sim \mathcal{D}_t(x, y))$, where $|L_s| = l_s \gg |L_t| = l_t$. We also have target unlabeled data denoted by $U_t(\sim \mathcal{D}_t(x))$, where $|U_t| = u_t$. Our goal is to learn a hypothesis $\mathbf{h} : \mathcal{X} \mapsto \mathcal{Y}$ having low expected error with respect to the target domain. In this paper, we consider *linear hypotheses* only. However, the proposed techniques extend to non-linear hypotheses, as mentioned in [1]. Source and target empirical errors for hypothesis $\mathbf{h}$ are denoted by $\hat{\epsilon}_s(\mathbf{h}, f_s)$ and $\hat{\epsilon}_t(\mathbf{h}, f_t)$ respectively, where $f_s$ and $f_t$ are the true source and target labeling functions. Similarly, the corresponding expected errors are denoted by $\epsilon_s(\mathbf{h}, f_s)$ and $\epsilon_t(\mathbf{h}, f_t)$. We will use shorthand notations of $\hat{\epsilon}_s, \hat{\epsilon}_t, \epsilon_s$ and $\epsilon_t$ wherever the intention is clear from context.

### 2.2 EasyAdapt (EA)

Let us denote $\mathbb{R}^d$ as the *original* space. EA operates in an *augmented* space denoted by $\breve{\mathcal{X}} \subset \mathbb{R}^{3d}$ (for a single pair of source and target domain). For $k$ domains, the *augmented* space blows up to $\mathbb{R}^{(k+1)d}$. The augmented feature maps $\Phi^s, \Phi^t : \mathcal{X} \mapsto \breve{\mathcal{X}}$ for source and target domains are defined as $\Phi^s(\mathbf{x}) = \langle \mathbf{x}, \mathbf{x}, \mathbf{0} \rangle$ and $\Phi^t(\mathbf{x}) = \langle \mathbf{x}, \mathbf{0}, \mathbf{x} \rangle$ where $\mathbf{x}$ and $\mathbf{0}$ are vectors in $\mathbb{R}^d$, and $\mathbf{0}$ denotes a zero vector of dimension $d$. The first $d$-dimensional segment corresponds to commonality between source and target, the second $d$-dimensional segment corresponds to the source domain while the last segment corresponds to the target domain. Source and target domain examples are transformed using these feature maps and the augmented features so constructed are passed onto the underlying supervised classifier. One of the most appealing properties of EASYADAPT is that it is agnostic of the underlying supervised classifier being used to learn in the *augmented* space. Almost any *standard supervised learning approach* (for e.g., SVMs, perceptrons) can be used to learn a *linear hypothesis* $\breve{\mathbf{h}} \in \mathbb{R}^{3d}$ in the augmented space. Let us denote $\breve{\mathbf{h}} = \langle \mathbf{g_c}, \mathbf{g_s}, \mathbf{g_t} \rangle$, where each of $\mathbf{g_c}$, $\mathbf{g_s}$, $\mathbf{g_t}$ is of dimension $d$, and represent the *common*, *source-specific* and *target-specific* components of $\breve{\mathbf{h}}$, respectively. During prediction on target data, the incoming target sample $\mathbf{x}$ is transformed to obtain $\Phi^t(\mathbf{x})$ and $\breve{\mathbf{h}}$ is applied on this transformed sample. This is equivalent to applying $(\mathbf{g_c} + \mathbf{g_t})$ on $\mathbf{x}$. A intuitive insight into why this simple algorithm works so well in practice and outperforms most state-of-the-art algorithms is given in [1]. Briefly, it can be thought to be simultaneously training two hypotheses: $\mathbf{h_s} = (\mathbf{g_c} + \mathbf{g_s})$ for source domain and $\mathbf{h_t} = (\mathbf{g_c} + \mathbf{g_t})$ for target domain. The commonality between the domains is represented by $\mathbf{g_c}$ whereas $\mathbf{g_s}$ and $\mathbf{g_t}$ capture the idiosyncrasies of the source and target domain, respectively.

## 3 EA++: EA using unlabeled data

As discussed in the previous section, the EASYADAPT algorithm is attractive because it performs very well empirically and can be used in conjunction with any underlying supervised *linear classifier*. One drawback of EASYADAPT is its inability to leverage unlabeled target data which is usually available in large quantities in most practical scenarios. In this section, we extend EA to semi-supervised settings while maintaining the desirable classifier-agnostic property.

### 3.1 Motivation

In multi-view approach to semi-supervised learning [12], different hypotheses are learned using different *views* of the dataset. Thereafter, unlabeled data is utilized to co-regularize these learned hypotheses by making them agree on unlabeled samples. In domain adaptation, the source and target data come from two different distributions. However, if the source and target domains are *reasonably close*, we can employ a similar form of regularization using unlabeled data. A prior co-regularization based idea to harness unlabeled data in domain adaptation tasks demonstrated improved empirical results [10]. However, their technique applies for the particular base classifier they consider and hence does not extend to other supervised classifiers.

### 3.2 EA++: EASYADAPT with unlabeled data

In our proposed semi-supervised approach, the source and target hypotheses are made to agree on unlabeled data. We refer to this algorithm as EA++. Recall that EASYADAPT learns a *linear hypothesis* $\breve{\mathbf{h}} \in \mathbb{R}^{3d}$ in the *augmented* space. The hypothesis $\breve{\mathbf{h}}$ contains common, source-specific and target-specific sub-hypotheses and is expressed as $\breve{\mathbf{h}} = \langle \mathbf{g_c}, \mathbf{g_s}, \mathbf{g_t} \rangle$. In *original* space (ref. Section 2.2), this is equivalent to learning a source specific hypothesis $\mathbf{h_s} = (\mathbf{g_c} + \mathbf{g_s})$ and a target specific hypothesis $\mathbf{h_t} = (\mathbf{g_c} + \mathbf{g_t})$.

In EA++, we want the source hypothesis $\mathbf{h_s}$ and the target hypothesis $\mathbf{h_t}$ to agree on the unlabeled data. For an unlabeled target sample $\mathbf{x}_i \in U_t \subset \mathbb{R}^d$, the goal of EA++ is to make the predictions of $\mathbf{h_s}$ and $\mathbf{h_t}$ on $\mathbf{x}_i$, agree with each other. Formally, it aims to achieve the following condition:

$$\mathbf{h_s} \cdot \mathbf{x_i} \approx \mathbf{h_t} \cdot \mathbf{x_i} \Longleftrightarrow (\mathbf{g_c} + \mathbf{g_s}) \cdot \mathbf{x_i} \approx (\mathbf{g_c} + \mathbf{g_t}) \cdot \mathbf{x_i}$$
$$\Longleftrightarrow (\mathbf{g_s} - \mathbf{g_t}) \cdot \mathbf{x_i} \approx 0 \Longleftrightarrow \langle \mathbf{g_c}, \mathbf{g_s}, \mathbf{g_t} \rangle \cdot \langle \mathbf{0}, \mathbf{x_i}, -\mathbf{x_i} \rangle \approx 0. \tag{3.1}$$

The above expression leads to the definition of a new feature map $\Phi^u : \mathcal{X} \mapsto \breve{\mathcal{X}}$ for unlabeled data given by $\Phi^u(\mathbf{x}) = \langle \mathbf{0}, \mathbf{x}, -\mathbf{x} \rangle$. Every unlabeled target sample is transformed using the map $\Phi^u(.)$. The augmented feature space that results from the application of three feature maps, namely, $\Phi^s(\cdot)$, $\Phi^t(\cdot)$ and $\Phi^u(\cdot)$ on source labeled samples, target labeled samples and target unlabeled samples is summarized in Figure 1(a).

As shown in Eq. 3.1, during the training phase, EA++ assigns a predicted value close to $0$ for each unlabeled sample. However, it is worth noting that during the test phase, EA++ predicts labels from two classes: $+1$ and $-1$. This warrants further exposition of the implementation specifics which is deferred until the next subsection.

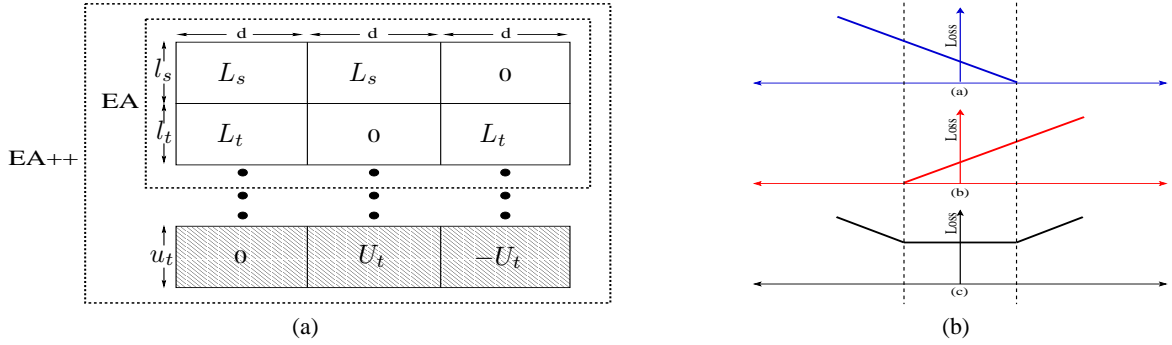

(a)                                                                                    (b)

Figure 1: (a) Diagrammatic representation of feature augmentation in EA and EA++, (b) Loss functions for class $+1$, class $-1$ and their summation.

### 3.3 Implementation

In this section, we present implementation specific details of EA++. For concreteness, we consider SVM as the base supervised learner. However, these details hold for other *supervised linear classifiers*. In the dual form of SVM optimization function, the labels are multiplied with features. Since, we want the predicted labels for unlabeled data to be $0$ (according to Eq. 3.1), multiplication by zero will make the unlabeled samples ineffective in the dual form of

the cost function. To avoid this, we create as many copies of $\Phi^u(\mathbf{x})$ as there are labels and assign each label to one copy of $\Phi^u(\mathbf{x})$. For the case of binary classification, we create two copies of every augmented unlabeled sample, and assign $+1$ label to one copy and $-1$ to the other. The learner attempts to balance the loss of the two copies, and tries to make the prediction on unlabeled sample equal to 0. Figure 1(b) shows the curves of the hinge loss for class $+1$, class $-1$ and their summation. The effective loss for each unlabeled sample is similar to the sum of losses for $+1$ and $-1$ classes (shown in Figure 1(b)c).

## 4  Generalization Bounds

In this section, we present Rademacher complexity based generalization bounds for EA and EA++. First, we define hypothesis classes for EA and EA++ using an alternate formulation. Second, we present a theorem (Theorem 4.1) which relates *empirical* and *expected* error for the general case and hence applies to both the source and target domains. Third, we prove Theorem 4.2 which relates the *expected target error* to the *expected source error*. Fourth, we present Theorem 4.3 which combines Theorem 4.1 and Theorem 4.2 so as to relate the *expected target error* to *empirical* errors in source and target (which is the main goal of the generalization bounds presented in this paper). Finally, all that remains is to bound the Rademacher complexity of the various hypothesis classes.

### 4.1  Define Hypothesis Classes for EA and EA++

Our goal now is to define the hypothesis classes for EA and EA++ so as to make the theoretical analysis feasible. Both EA and EA++ train hypotheses in the augmented space $\breve{\mathcal{X}} \subset \mathbb{R}^{3d}$. The augmented hypothesis $\breve{h}$ is trained using data from both domains, and the three sub-hypotheses $(\mathbf{g_c} + \mathbf{g_s} + \mathbf{g_t})$ of $d$-dimension are treated in a different manner for source and target data. We use an alternate formulation of the hypothesis classes and work in the original space $\mathcal{X} \subset \mathbb{R}^d$. As discussed briefly in Section 2.2, EA can be thought to be simultaneously training two hypotheses $\mathbf{h_s} = (\mathbf{g_c} + \mathbf{g_s})$ and $\mathbf{h_t} = (\mathbf{g_c} + \mathbf{g_t})$ for source and target domains, respectively. We consider the case when the underlying supervised classifier in augmented space uses a square $L_2$-norm regularizer of the form $||\breve{h}||^2$ (as used in SVM). This is equivalent to imposing the regularizer $(||\mathbf{g_c}||^2 + ||\mathbf{g_s}||^2 + ||\mathbf{g_t}||^2) = (||\mathbf{g_c}||^2 + ||\mathbf{h_s} - \mathbf{g_c}||^2 + ||\mathbf{h_t} - \mathbf{g_c}||^2)$. Differentiating this regularizer w.r.t. $\mathbf{g_c}$ gives $\mathbf{g_c} = (\mathbf{h_s} + \mathbf{h_t})/3$ at the minimum, and the regularizer reduces to $\frac{1}{3}(||\mathbf{h_s}||^2 + ||\mathbf{h_t}||^2 + ||\mathbf{h_s} - \mathbf{h_t}||^2)$. Thus, EA can be thought to be minimizing the sum of empirical source error on $\mathbf{h_s}$, empirical target error on $\mathbf{h_t}$ and this regularizer. The cost function $\mathcal{Q}_{EA}(\mathbf{h_1}, \mathbf{h_2})$ can now be written as:

$$\alpha\hat{\epsilon}_s(\mathbf{h_1}) + (1-\alpha)\hat{\epsilon}_t(\mathbf{h_2}) + \lambda_1||\mathbf{h_1}||^2 + \lambda_2||\mathbf{h_2}||^2 + \lambda||\mathbf{h_1} - \mathbf{h_2}||^2, \quad \text{and} \quad (\mathbf{h_s}, \mathbf{h_t}) = \underset{\mathbf{h_1}, \mathbf{h_2}}{\arg\min}\,\mathcal{Q}_{EA} \quad (4.1)$$

The EA algorithm minimizes this cost function over $\mathbf{h_1}$ and $\mathbf{h_2}$ jointly to obtain $\mathbf{h_s}$ and $\mathbf{h_t}$. The EA++ algorithm uses target unlabeled data, and encourages $\mathbf{h_s}$ and $\mathbf{h_t}$ to agree on unlabeled samples (Eq. 3.1). This can be thought of as having an additional regularizer of the form $\sum_{i \in U_t}(\mathbf{h_s}(x_i) - \mathbf{h_t}(x_i))^2$ in the cost function. The cost function for EA++ (denoted as $\mathcal{Q}_{++}(\mathbf{h_1}, \mathbf{h_2})$) can then be written as:

$$\alpha\hat{\epsilon}_s(\mathbf{h_1}) + (1-\alpha)\hat{\epsilon}_t(\mathbf{h_2}) + \lambda_1||\mathbf{h_1}||^2 + \lambda_2||\mathbf{h_2}||^2 + \lambda||\mathbf{h_1} - \mathbf{h_2}||^2 + \lambda_u\sum_{i \in U_t}(\mathbf{h_1}(x_i) - \mathbf{h_2}(x_i))^2 \quad (4.2)$$

Both EA and EA++ give equal weights to source and target empirical errors, so $\alpha$ turns out to be 0.5. We use hyperparameters $\lambda_1$, $\lambda_2$, $\lambda$, and $\lambda_u$ in the cost functions to make them more general. However, as explained earlier, EA implicitly sets all these hyperparameters $(\lambda_1, \lambda_2, \lambda)$ to the same value (which will be $0.5(\frac{1}{3}) = \frac{1}{6}$ in our case, since the weights in the entire cost function are multiplied by $\alpha = 0.5$). The hyperparameter for unlabeled data $(\lambda_u)$ is 0.5 in EA++. We assume that the loss $L(y, \mathbf{h.x})$ is bounded by 1 for the zero hypothesis $\mathbf{h} = \mathbf{0}$. This is true for many popular loss functions including square loss and hinge loss when $y \in \{-1, +1\}$. One possible way [13] of defining the hypotheses classes is to substitute trivial hypotheses $\mathbf{h_1} = \mathbf{h_2} = 0$ in both the cost functions which makes all regularizers and co-regularizers equal to zero and thus bounds the cost functions $\mathcal{Q}_{EA}$ and $\mathcal{Q}_{++}$. This gives us $\mathcal{Q}_{EA}(\mathbf{0}, \mathbf{0}) \leq 1$ and $\mathcal{Q}_{++}(\mathbf{0}, \mathbf{0}) \leq 1$ since $\hat{\epsilon}_s(\mathbf{0}), \hat{\epsilon}_t(\mathbf{0}) \leq 1$. Without loss of generality, we also assume that final source and target hypotheses can only reduce the cost function as compared to the zero hypotheses. Hence, the final hypothesis pair $(\mathbf{h_s}, \mathbf{h_t})$ that minimizes the cost functions is contained in the following paired hypothesis classes for EA and EA++,

$$\mathcal{H} := \{(\mathbf{h_1}, \mathbf{h_2}) : \lambda_1||\mathbf{h_1}||^2 + \lambda_2||\mathbf{h_2}||^2 + \lambda||\mathbf{h_1} - \mathbf{h_2}||^2 \leq 1\}$$

$$\mathcal{H}_{++} := \{(\mathbf{h_1}, \mathbf{h_2}) : \lambda_1||\mathbf{h_1}||^2 + \lambda_2||\mathbf{h_2}||^2 + \lambda||\mathbf{h_1} - \mathbf{h_2}||^2 + \lambda_u\sum_{i \in U_t}(\mathbf{h_1}(x_i) - \mathbf{h_2}(x_i))^2 \leq 1\} \quad (4.3)$$

The source hypothesis class for EA is the set of all $h_1$ such that the pair $(h_1, h_2)$ is in $\mathcal{H}$. Similarly, the target hypothesis class for EA is the set of all $h_2$ such that the pair $(h_1, h_2)$ is in $\mathcal{H}$. Consequently, the source and target hypothesis classes for EA can be defined as:

$$\mathcal{J}^s_{EA} := \{\mathbf{h_1} : \mathcal{X} \mapsto \mathbb{R}, (\mathbf{h_1}, \mathbf{h_2}) \in \mathcal{H}\} \qquad \text{and} \qquad \mathcal{J}^t_{EA} := \{\mathbf{h_2} : \mathcal{X} \mapsto \mathbb{R}, (\mathbf{h_1}, \mathbf{h_2}) \in \mathcal{H}\} \qquad (4.4)$$

Similarly, the source and target hypothesis classes for EA++ are defined as:

$$\mathcal{J}^s_{++} := \{\mathbf{h_1} : \mathcal{X} \mapsto \mathbb{R}, (\mathbf{h_1}, \mathbf{h_2}) \in \mathcal{H}_{++}\} \qquad \text{and} \qquad \mathcal{J}^t_{++} := \{\mathbf{h_2} : \mathcal{X} \mapsto \mathbb{R}, (\mathbf{h_1}, \mathbf{h_2}) \in \mathcal{H}_{++}\} \qquad (4.5)$$

Furthermore, we assume that our hypothesis class is comprised of real-valued functions over an RKHS with reproducing kernel $k(\cdot, \cdot)$, $k : \mathcal{X} \times \mathcal{X} \mapsto \mathbb{R}$. Let us define the kernel matrix and partition it corresponding to source labeled, target labeled and target unlabeled data as shown below:

$$K = \begin{pmatrix} A_{s \times s} & C_{s \times t} & D_{s \times u} \\ C'_{t \times s} & B_{t \times t} & E_{t \times u} \\ D'_{u \times s} & E'_{u \times t} & F_{u \times u} \end{pmatrix}, \qquad (4.6)$$

where 's', 't' and 'u' indicate terms corresponding to source labeled, target labeled and target unlabeled, respectively.

## 4.2 Relate empirical and expected error (for both source and target)

Having defined the hypothesis classes, we now proceed to obtain generalization bounds for EA and EA++. We have the following standard generalization bound based on the Rademacher complexity of a hypothesis class [13].

**Theorem 4.1.** *Suppose the uniform Lipschitz condition holds for $L : \mathcal{Y}^2 \to [0, 1]$, i.e., $|L(\hat{y}_1, y) - L(\hat{y}_2, y)| \leq M|\hat{y}_1 - \hat{y}_2|$, where $y, \hat{y}_1, \hat{y}_2 \in \mathcal{Y}$ and $\hat{y}_1 \neq \hat{y}_2$. Then for any $\delta \in (0, 1)$ and for $m$ samples $(X_1, Y_1), (X_2, Y_2), \ldots, (X_m, Y_m)$ drawn i.i.d. from distribution $\mathcal{D}$, we have with probability at least $(1 - \delta)$ over random draws of samples,*

$$\epsilon(f) \leq \hat{\epsilon}(f) + 2M\hat{R}_m(\mathcal{F}) + \frac{1}{\sqrt{m}}(2 + 3\sqrt{ln(2/\delta)/2}).$$

*where $f \in \mathcal{F}$ is the class of functions mapping $\mathcal{X} \mapsto \mathcal{Y}$, and $\hat{R}_m(\mathcal{F})$ is the empirical Rademacher complexity of $\mathcal{F}$ defined as $\hat{R}_m(\mathcal{F}) := E_\sigma[\sup_{f \in \mathcal{F}} |\frac{2}{m} \sum_{i=1}^m \sigma_i h_2(x_i)|]$.*

If we can bound the complexity of hypothesis classes $\mathcal{J}^s_{EA}$ and $\mathcal{J}^t_{EA}$, we will have a uniform convergence bound on the difference of expected and empirical errors ($|\epsilon_t(h) - \hat{\epsilon}_t(h)|$ and $|\epsilon_s(h) - \hat{\epsilon}_s(h)|$) using Theorem 4.1. However, in domain adaptation setting, we are also interested in the bounds that relate expected target error to total empirical error on source and target samples. The following sections aim to achieve this goal.

## 4.3 Relate source expected error and target expected error

The following theorem provides a bound on the difference of expected target error and expected source error. The bound is in terms of $\eta_s := \epsilon_s(f_s, f_t)$, $\nu_s := \epsilon_s(h_t^*, f_t)$ and $\nu_t := \epsilon_t(h_t^*, f_t)$, where $f_s$ and $f_t$ are the source and target labeling functions, and $h_t^*$ is the optimal target hypothesis in target hypothesis class. It also uses $d_{\mathcal{H}\Delta\mathcal{H}}(\mathcal{D}_s, \mathcal{D}_t)-$ distance [14], which is defined as $\sup_{h_1, h_2 \in \mathcal{H}} 2|\epsilon_s(h_1, h_2) - \epsilon_t(h_1, h_2)|$. The $d_{\mathcal{H}\Delta\mathcal{H}}-$distance measures the distance between two distribution using a hypothesis class-specific distance measure. If the two domains are close to each other, $\eta_s$ and $d_{\mathcal{H}\Delta\mathcal{H}}(\mathcal{D}_s, \mathcal{D}_t)$ are expected to be small. On the contrary, if the domains are far apart, these terms will be big and the use of extra source samples may not help in learning a better target hypothesis. These two terms also represent the notion of adaptability in our case.

**Theorem 4.2.** *Suppose the loss function is M-Lipschitz as defined in Theorem 4.1, and obeys triangle inequality. For any two source and target hypotheses $h_s, h_t$ (which belong to different hypotheses classes), we have*

$$\epsilon_t(h_t, f_t) - \epsilon_s(h_s, f_s) \leq M||h_t - h_s||E_s\left[\sqrt{k(x, x)}\right] + \frac{1}{2}d_{\mathcal{H}_t\Delta\mathcal{H}_t}(D_s, D_t) + \eta_s + \nu_s + \nu_t.$$

*where $\mathcal{H}_t$ is the target hypothesis class, and $k(\cdot, \cdot)$ is the reproducing kernel for the RKHS. $\eta_s$, $\nu_s$, and $\nu_t$ are defined as above.*

*Proof.* Please see Appendix A in the supplement. $\qquad\qquad\square$

### 4.4 Relate target expected error with source and target empirical errors

EA and EA++ learn source and target hypotheses jointly. So the empirical error in one domain is expected to have its effect on the generalization error in the other domain. In this section, we aim to bound the target expected error in terms of source and target empirical errors. The following theorem achieves this goal.

**Theorem 4.3.** *Under the assumptions and definitions used in Theorem 4.1 and Theorem 4.2, with probability at least $1 - \delta$ we have*

$$\epsilon_t(h_t, f_t) \leq \frac{1}{2}(\hat{\epsilon}_s(h_s, f_s) + \hat{\epsilon}_t(h_t, f_t)) + \frac{1}{2}(2M\hat{R}_m(\mathcal{H}_s) + 2M\hat{R}_m(\mathcal{H}_t)) + \frac{1}{2}\left(\frac{1}{\sqrt{l_s}} + \frac{1}{\sqrt{l_t}}\right)(2 + 3\sqrt{ln(2/\delta)/2})$$
$$+ \frac{1}{2}M||h_t - h_s||E_s\left[\sqrt{k(x,x)}\right] + \frac{1}{4}d_{\mathcal{H}_t \Delta \mathcal{H}_t}(D_s, D_t) + \frac{1}{2}(\eta_s + \nu_s + \nu_t)$$

*for any $h_s$ and $h_t$. $\mathcal{H}_s$ and $\mathcal{H}_t$ are the source hypothesis class and the target hypothesis class, respectively.*

*Proof.* We first use Theorem 4.1 to bound $(\epsilon_t(h_t) - \hat{\epsilon}_t(h_t))$ and $(\epsilon_s(h_s) - \hat{\epsilon}_s(h_s))$. The above theorem directly follows by combining these two bounds and Theorem 4.2. $\qquad\square$

This bound provides better a understanding of how the target expected error is governed by both source and target empirical errors, and hypotheses class complexities. This behavior is expected since both EA and EA++ learn source and target hypotheses jointly. We also note that the bound in Theorem 4.3 depends on $||h_s - h_t||$, which apparently might give an impression that the best possible thing to do is to make source and target hypotheses equal. However, due to joint learning of source and target hypotheses (by optimizing the cost function of Eq. 4.1), making the source and target hypotheses close will increase the source empirical error, thus loosening the bound of Theorem 4.3. Noticing that $||h_s - h_t||^2 \leq \frac{1}{\lambda}$ for both EA and EA++, the bound can be made independent of $||h_s - h_t||$ although with a sacrifice on the tightness. We note that Theorem 4.1 can also be used to bound the target generalization error of EA and EA++ in terms of only target empirical error. However, if the number of labeled target samples is extremely low, this bound can be loose due to inverse dependency on number of target samples. Theorem 4.3 bounds the target expected error using the averages of empirical errors, Rademacher complexities, and sample dependent terms. If the domains are reasonably close and the number of labeled source samples is much higher than target samples, this can provide a tighter bound compared to Theorem 4.1.

Finally, we need the Rademacher complexities of source and target hypothesis classes (for both EA and EA++) to be able to use Theorem 4.3, which are provided in the next sections.

### 4.5 Bound the Complexity of EA and EA++ Hypothesis Classes

The following theorems bound the Rademacher complexity of the target hypothesis classes for EA and EA++.

#### 4.5.1 EASYADAPT (EA)

**Theorem 4.4.** *For the hypothesis class $\mathcal{J}_{EA}^t$ defined in Eq. 4.4 we have, $\frac{1}{\sqrt[4]{2}}\frac{2C_{EA}^t}{l_t} \leq \hat{R}_m(\mathcal{J}_{EA}^t) \leq \frac{2C_{EA}^t}{l_t}$ where, $\hat{R}_m(\mathcal{J}_{EA}^t) = E_\sigma \sup_{h_2 \in \mathcal{J}_{EA}^t} |\sum_i \sigma_i h_2(x)|$, $(C_{EA}^t)^2 = \left(\frac{1}{\lambda_2 + \left(\frac{1}{\lambda_1} + \frac{1}{\lambda}\right)^{-1}}\right) tr(B)$ and $B$ is the kernel sub-matrix defined as in Eq. 4.6.*

*Proof.* Please see Appendix B in the supplement. $\qquad\square$

The complexity of target class decreases with an increase in the values of hyperparameters. It decreases more rapidly with change in $\lambda_2$ compared to $\lambda$ and $\lambda_1$, which is also expected since $\lambda_2$ is the hyperparameter directly influencing the target hypothesis. The kernel block sub-matrix corresponding to source samples does not appear in the bound. This result in conjunction with Theorem 4.1 gives a bound on the target generalization error.

To be able to use the bound of Theorem 4.3, we need the Rademacher complexity of the source hypothesis class. Due to the symmetry of paired hypothesis class (Eq. 4.3) in $h_1$ and $h_2$ up to scalar parameters, the complex-

ity of source hypothesis class can be similarly bounded by $\frac{1}{\sqrt[4]{2}} \frac{2C_{EA}^s}{l_s} \leq \hat{R}_m(\mathcal{J}_{EA}^s) \leq \frac{2C_{EA}^s}{l_s}$, where $(C_{EA}^s)^2 = \left( \frac{1}{\lambda_1 + \left( \frac{1}{\lambda_2} + \frac{1}{\lambda} \right)^{-1}} \right) tr(A)$, and $A$ is the kernel block sub-matrix corresponding to source samples.

### 4.5.2 EASYADAPT++ (EA++)

**Theorem 4.5.** *For the hypothesis class $\mathcal{J}_{++}^t$ defined in Eq. 4.5 we have,* $\frac{1}{\sqrt[4]{2}} \frac{2C_{++}^t}{l_t} \leq \hat{R}_m(\mathcal{J}_{++}^t) \leq \frac{2C_{++}^t}{l_t}$ *where,* $\hat{R}_m(\mathcal{J}_{++}^t) = E_\sigma \sup_{h_2 \in \mathcal{J}_{++}^t} |\sum_i \sigma_i h_2(x)|$ *and* $(C_{++}^t)^2 = \left( \frac{1}{\lambda_2 + \left( \frac{1}{\lambda_1} + \frac{1}{\lambda} \right)^{-1}} \right) tr(B) - \lambda_u \left( \frac{\lambda_1}{\lambda \lambda_1 + \lambda \lambda_2 + \lambda_1 \lambda_2} \right)^2 tr\left( E(I + kF)^{-1} E' \right)$, *where* $k = \frac{\lambda_u(\lambda_1 + \lambda_2)}{\lambda \lambda_1 + \lambda \lambda_2 + \lambda_1 \lambda_2}$.

*Proof.* Please see Appendix C in the Supplement. □

The second term in $(C_{++}^t)^2$ is always positive since the trace of a positive definite matrix is positive. So, the unlabeled data results in a reduction of complexity over the labeled data case (Theorem 4.4). The *trace* term in the reduction can also be written as $\sum_i ||E_i||^2_{(I+kF)^{-1}}$, where $E_i$ is the $i$'th column of matrix $E$ and $|| \cdot ||^2_Z$ is the norm induced by a positive definite matrix $Z$. Since $E_i$ is the vector representing the inner product of $i$'th target sample with all unlabeled samples, this means that the reduction in complexity is proportional to the *similarity* between target unlabeled samples and target labeled samples. This result in conjunction with Theorem 4.1 gives a bound on the target generalization error in terms of target empirical error.

To be able to use the bound of Theorem 4.3, we need the Rademacher complexity of source hypothesis class too. Again, as in case of EA, using the symmetry of paired hypothesis class $\mathcal{H}_{++}$ (Eq. 4.3) in $h_1$ and $h_2$ up to scalar parameters, the complexity of source hypothesis class can be similarly bounded by $\frac{1}{\sqrt[4]{2}} \frac{2C_{++}^s}{l_s} \leq \hat{R}_m(\mathcal{J}_{++}^s) \leq \frac{2C_{++}^s}{l_s}$, where $(C_{++}^s)^2 = \left( \frac{1}{\lambda_1 + \left( \frac{1}{\lambda_2} + \frac{1}{\lambda} \right)^{-1}} \right) tr(A) - \lambda_u \left( \frac{\lambda_2}{\lambda \lambda_1 + \lambda \lambda_2 + \lambda_1 \lambda_2} \right)^2 tr\left( D(I + kF)^{-1} D' \right)$, and $k$ is defined similarly as in Theorem 4.5. The *trace* term can again be interpreted as before, which implies that the reduction in source class complexity is proportional to the *similarity* between source labeled samples and target unlabeled samples.

## 5 Experiments

We follow experimental setups similar to [1] but report our empirical results for the task of sentiment classification using the SENTIMENT data provided by [15]. The task of sentiment classification is a binary classification task which corresponds to classifying a review as positive or negative for user reviews of eight product types (apparel, books, DVD, electronics, kitchen, music, video, and other) collected from *amazon.com*. We quantify the domain divergences in terms of the $\mathcal{A}$-distance [16] which is computed [17] from finite samples of source and target domain using the *proxy* $\mathcal{A}$-distance [16]. For our experiments, we consider the following domain-pairs: (a) DVD→BOOKS (*proxy* $\mathcal{A}$-distance=0.7616) and, (b) KITCHEN→APPAREL (*proxy* $\mathcal{A}$-distance=0.0459). As in [1], we use an averaged perceptron classifier from the Megam framework (implementation due to [18]) for all the aforementioned tasks. The training sample size varies from $1k$ to $16k$. In all cases, the amount of unlabeled target data is equal to the total amount of labeled source and target data.

We compare the empirical performance of EA++ with a few other baselines, namely, (a) SOURCEONLY (classifier trained on source labeled samples), (b) TARGETONLY-FULL (classifier trained on the same number of target labeled samples as the number of source labeled samples in SOURCEONLY), (c) TARGETONLY (classifier trained on small amount of target labeled samples, roughly one-tenth of the amount of source labeled samples in SOURCEONLY), (d) ALL (classifier trained on combined labeled samples of SOURCEONLY and TARGETONLY), (e) EA (classifier trained in *augmented feature space* on the same input training set as ALL), (f) EA++ (classifier trained in *augmented feature space* on the same input training set as EA and an equal amount of unlabeled *target* data). All these approaches were tested on the entire amount of available *target* test data.

Figure 2 presents the learning curves for (a) SOURCEONLY, (b) TARGETONLY-FULL, (c) TARGETONLY, (d) ALL, (e) EA, and (f) EA++ (EA with unlabeled data). The x-axis represents the number of training samples on which the

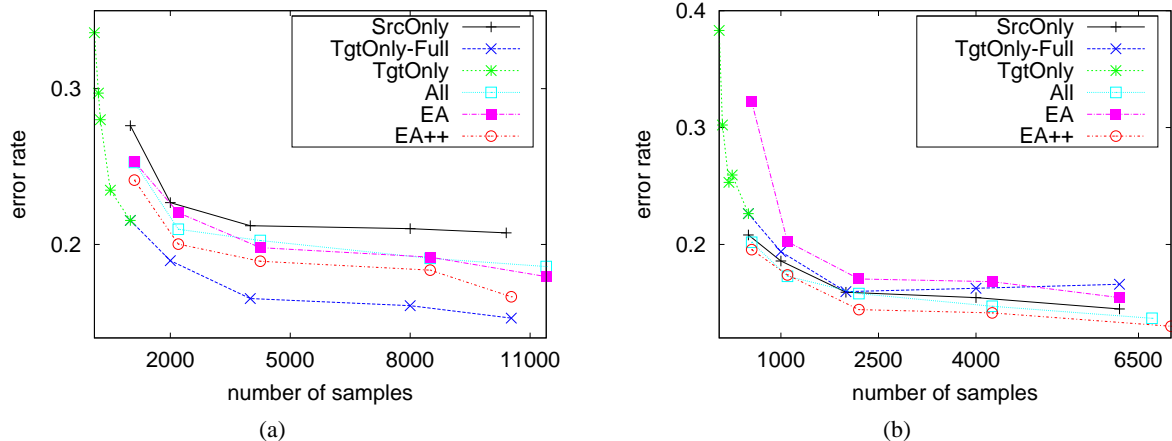

Figure 2: Test accuracy of SOURCEONLY, TARGETONLY-FULL, TARGETONLY, ALL, EA, EA++ (with unlabeled data) for, (a) DVD→BOOKS (*proxy $\mathcal{A}$-distance*=0.7616), (b) KITCHEN→APPAREL (*proxy $\mathcal{A}$-distance*=0.0459)

predictor has been trained. At this point, we note that the number of training samples vary depending on the particular approach being used. For SOURCEONLY, TARGETONLY-FULL and TARGETONLY, it is just the corresponding number of labeled source or target samples, respectively. For ALL and EA, it is the summation of labeled source and target samples. For EA++, the $x$-value plotted denotes the amount of unlabeled target data used (in addition to an equal amount of source+target labeled data, as in ALL or EA). We plot this number for EA++, just to compare its improvement over EA when using an additional (and equal) amount of unlabeled target data. This accounts for the different $x$ values plotted for the different curves. In all cases, the y-axis denotes the error rate.

As can be seen, for both the cases, EA++ outperforms EASYADAPT. For DVD→BOOKS, the domains are far apart as denoted by a high *proxy $\mathcal{A}$-distance*. Hence, TARGETONLY-FULL achieves the best performance and EA++ almost catches up for large amounts of training data. For different number of sample points, EA++ gives relative improvements in the range of $4.36\% - 9.14\%$, as compared to EA. The domains KITCHEN and APPAREL can be considered to be reasonably close due to their low domain divergence. Hence, this domain pair is more amenable for domain adaptation as is demonstrated by the fact that the other approaches (SOURCEONLY, TARGETONLY, ALL) perform better or atleast as good as TARGETONLY-FULL. However, as earlier, EA++ once again outperforms all these approaches including TARGETONLY-FULL. Due to the closeness of the two domains, additional unlabeled data in EA++ helps it in outperforming TARGETONLY-FULL. At this point, we also note that EA performs poorly for some cases, which corroborates with prior experimental results [1]. For this dataset, EA++ yields relative improvements in the range of $14.08\% - 39.29\%$ over EA for different number of sample points experimented with. Similar trends were observed for other tasks and datasets (refer Figure 3 of [2]).

## 6  Conclusions

We proposed a semi-supervised extension to an existing domain adaptation technique (EA). Our approach EA++, leverages unlabeled data to improve the performance of EA. With this extension, EA++ applies to both *fully supervised* and *semi-supervised* domain adaptation settings. We have formulated EA and EA++ in terms of co-regularization, an idea that originated in the context of multiview learning [13, 19]. Our proposed formulation also bears resemblance to existing work [20] in semi-supervised (SSL) literature which has been studied extensively in [21, 22, 23]. The difference being, while in SSL one would try to make the two views (on unlabeled data) agree, in domain adaptation the aim is to make the two hypotheses in source and target agree. Using our formulation, we have presented theoretical analysis of the superior performance of EA++ as compared to EA. Our empirical results further confirm the theoretical findings. EA++ can also be extended to the multiple source settings. If we have $k$ sources and a single target domain then we can introduce a co-regularizer for each source-target pair. Due to space constraints, we defer details to a full version.

## Footnotes

[1] A preliminary version [2] of this work appeared in the DANLP workshop at ACL 2010.

[2] We define *supervised domain adaptation* as having labeled data in both *source* and *target* and *unsupervised domain adaptation* as having labeled data in only *source*. In *semi-supervised domain adaptation*, we also have access to both labeled and unlabeled data in *target*.

# References

[1] Hal Daumé III. Frustratingly easy domain adaptation. In *ACL'07*, pages 256–263, Prague, Czech Republic, June 2007.

[2] Hal Daumé III, Abhishek Kumar, and Avishek Saha. Frustratingly easy semi-supervised domain adaptation. In *ACL 2010 Workshop on Domain Adaptation for Natural Language Processing (DANLP)*, pages 53–59, Uppsala, Sweden, July 2010.

[3] Theodoros Evgeniou and Massimiliano Pontil. Regularized multitask learning. In *KDD'04*, pages 109–117, Seattle, WA, USA, August 2004.

[4] Mark Dredze, Alex Kulesza, and Koby Crammer. Multi-domain learning by confidence-weighted parameter combination. *Machine Learning*, 79(1-2):123–149, 2010.

[5] Andrew Arnold and William W. Cohen. Intra-document structural frequency features for semi-supervised domain adaptation. In *CIKM'08*, pages 1291–1300, Napa Valley, California, USA, October 2008.

[6] John Blitzer, Ryan Mcdonald, and Fernando Pereira. Domain adaptation with structural correspondence learning. In *EMNLP'06*, pages 120–128, Sydney, Australia, July 2006.

[7] Gokhan Tur. Co-adaptation: Adaptive co-training for semi-supervised learning. In *ICASSP'09*, pages 3721–3724, Taipei, Taiwan, April 2009.

[8] Wenyuan Dai, Gui-Rong Xue, Qiang Yang, and Yong Yu. Transferring Naive Bayes classifiers for text classification. In *AAAI'07*, pages 540–545, Vancouver, B.C., July 2007.

[9] Dikan Xing, Wenyuan Dai, Gui-Rong Xue, and Yong Yu. Bridged refinement for transfer learning. In *PKDD'07*, pages 324–335, Warsaw, Poland, September 2007.

[10] Lixin Duan, Ivor W. Tsang, Dong Xu, and Tat-Seng Chua. Domain adaptation from multiple sources via auxiliary classifiers. In *ICML'09*, pages 289–296, Montreal, Quebec, June 2009.

[11] Ming-Wei Chang, Michael Connor, and Dan Roth. The necessity of combining adaptation methods. In *EMNLP'10*, pages 767–777, Cambridge, MA, October 2010.

[12] Vikas Sindhwani, Partha Niyogi, and Mikhail Belkin. A co-regularization approach to semi-supervised learning with multiple views. In *ICML Workshop on Learning with Multiple Views*, pages 824–831, Bonn, Germany, August 2005.

[13] D. S. Rosenberg and P. L. Bartlett. The Rademacher complexity of co-regularized kernel classes. In *AISTATS'07*, pages 396–403, San Juan, Puerto Rico, March 2007.

[14] John Blitzer, Koby Crammer, Alex Kulesza, Fernando Pereira, and Jennifer Wortman. Learning bounds for domain adaptation. In *NIPS'07*, pages 129–136, Vancouver, B.C., December 2007.

[15] John Blitzer, Mark Dredze, and Fernando Pereira. Biographies, bollywood, boom-boxes and blenders: Domain adaptation for sentiment classification. In *ACL'07*, pages 440–447, Prague, Czech Republic, June 2007.

[16] Shai Ben-David, John Blitzer, Koby Crammer, and Fernando Pereira. Analysis of representations for domain adaptation. In *NIPS'06*, pages 137–144, Vancouver, B.C., December 2006.

[17] Piyush Rai, Avishek Saha, Hal Daumé III, and Suresh Venkatasubramanian. Domain adaptation meets active learning. In *NAACL 2010 Workshop on Active Learning for NLP (ALNLP)*, pages 27–32, Los Angeles, USA, June 2010.

[18] Hal Daumé III. Notes on CG and LM-BFGS optimization of logistic regression. August 2004.

[19] Vikas Sindhwani and David S. Rosenberg. An RKHS for multi-view learning and manifold co-regularization. In *ICML'08*, pages 976–983, Helsinki, Finland, June 2008.

[20] Avrim Blum and Tom Mitchell. Combining labeled and unlabeled data with co-training. In *COLT'98*, pages 92–100, New York, NY, USA, July 1998. ACM.

[21] Maria-Florina Balcan and Avrim Blum. A PAC-style model for learning from labeled and unlabeled data. In *COLT'05*, pages 111–126, Bertinoro, Italy, June 2005.

[22] Maria-Florina Balcan and Avrim Blum. A discriminative model for semi-supervised learning. *J. ACM*, 57(3), 2010.

[23] Karthik Sridharan and Sham M. Kakade. An information theoretic framework for multi-view learning. In *COLT'08*, pages 403–414, Helsinki, Finland, June 2008.

